# Sparse Greedy Minimax Probability Machine Classification

**Thomas R. Strohmann**
Department of Computer Science
University of Colorado, Boulder
*strohman@cs.colorado.edu*

**Andrei Belitski**
Department of Computer Science
University of Colorado, Boulder
*Andrei.Belitski@colorado.edu*

**Gregory Z. Grudic**
Department of Computer Science
University of Colorado, Boulder
*grudic@cs.colorado.edu*

**Dennis DeCoste**
Machine Learning Systems Group
NASA Jet Propulsion Laboratory
*decoste@aig.jpl.nasa.gov*

## Abstract

The Minimax Probability Machine Classification (MPMC) framework [Lanckriet *et al.*, 2002] builds classifiers by minimizing the maximum probability of misclassification, and gives direct estimates of the probabilistic accuracy bound $\Omega$. The only assumptions that MPMC makes is that good estimates of means and covariance matrices of the classes exist. However, as with Support Vector Machines, MPMC is computationally expensive and requires extensive cross validation experiments to choose kernels and kernel parameters that give good performance. In this paper we address the computational cost of MPMC by proposing an algorithm that constructs nonlinear *sparse* MPMC (SMPMC) models by incrementally adding basis functions (i.e. kernels) one at a time – greedily selecting the next one that maximizes the accuracy bound $\Omega$. SMPMC automatically chooses both kernel parameters and feature weights *without* using computationally expensive cross validation. Therefore the SMPMC algorithm simultaneously addresses the problem of kernel selection and feature selection (i.e. feature weighting), based solely on maximizing the accuracy bound $\Omega$. Experimental results indicate that we can obtain reliable bounds $\Omega$, as well as test set accuracies that are comparable to state of the art classification algorithms.

## 1   Introduction

The goal of a binary classifier is to maximize the probability that unseen test data will be classified correctly. Assuming that the test data is generated from the same probability distribution as the training data, it is possible to derive specific probability bounds for the case that the decision boundary is a hyperplane. The following result due to Marshall and Olkin [1] and extended by Bertsimas and Popescu [2] provides the theoretical basis for

assigning probability bounds to hyperplane classifiers:

$$\sup_{E[\mathbf{z}]=\bar{\mathbf{z}},Cov[\mathbf{z}]=\Sigma_\mathbf{z}} Pr\{\mathbf{a}^T\mathbf{z} \geq b\} = \frac{1}{1+\omega^2} \quad \omega^2 = \inf_{\mathbf{a}^T\mathbf{t}\geq b}(\mathbf{t}-\bar{\mathbf{z}})^T\Sigma_\mathbf{z}^{-1}(\mathbf{t}-\bar{\mathbf{z}}) \quad (1)$$

where $\mathbf{a} \in \mathbf{R}^d, b$ are the hyperplane parameters, $\mathbf{z}$ is a random vector, and $\mathbf{t}$ is an ordinary vector. Lanckriet et al (see [3] and [4]) used the above result to build the Minimax Probability Machine for binary classification (MPMC). From (1) we note that the only required relevant information of the underlying probability distribution for each class is its mean and covariance matrix. No other estimates and/or assumptions are needed, which implies that the obtained bound (which we refer to as $\Omega$) is essentially distribution free, i.e. it holds for *any* distribution with a certain mean and covariance matrix.

As with other classification algorithms such as Support Vector Machines (SVM) (see [5]), the main disadvantage of current MPMC implementations is that they are computationally expensive (same complexity as SVM), and require extensive cross validation experiments to choose kernels and kernel parameter to give good performance on each data set. The goal of this paper is to propose a kernel based MPMC algorithm that directly addresses these computational issues.

Towards this end, we propose a sparse greedy MPMC (SMPMC) algorithm that efficiently builds classifiers, while at the same time maintains the distribution free probability bound of MPM type algorithms. To achieve this goal, we propose to use an iterative algorithm which adds basis functions (i.e. kernels) one by one, to an initially "empty" model. We are considering basis functions that are induced by Mercer kernels, i.e. functions of the following form $f(\mathbf{z}) = K_\gamma(\mathbf{z}, \mathbf{z_i})$ (where $\mathbf{z_i}$ is an input vector of the training data). Bases are added in a greedy way: we select the particular $\mathbf{z_i}$ that maximizes the MPMC objective $\Omega$. Furthermore, SMPMC chooses optimal kernel parameters that maximize this metric (hence the subscript $\gamma$ in $K_\gamma$), including automatically weighting input features by $\gamma_j \geq 0$ for each kernel added, such that $\mathbf{z_i} = (\gamma_1 z_1, \gamma_2 z_2, ..., \gamma_d z_d)$ for $d$ dimensional data. The proposed SMPMC algorithm automatically selects kernels and re-weights features (i.e. does feature selection) for each new added basis function, by minimizing the error bound (i.e. maximizing $\Omega$). Thus the large computational cost of cross validation (typically used by SVM and MPMC) is avoided.

The paper is organized as follows: Section 2.1 reviews the standard MPMC; Section 2.2 describes the proposed sparse greedy MPMC algorithm (SMPMC); and Sections 2.3-2.4 show how we can use sparse MPMC to determine optimal kernel parameters. In section 3 we compare our results to the ones described in the original MPMC paper (see [4]), showing the probability bounds and the test set accuracies for different binary classification problems. The conclusion is presented in section 4. Matlab source code for the SMPMC algorithm is available online: `http://nago.cs.colorado.edu/~strohman/papers.html`

## 2 Classification model

In this section we develop a sparse version of the Minimax Probability Machine for binary classification. We show that besides a significant reduction in computational cost, the SMPMC algorithm allows us to do automated kernel and feature selection.

### 2.1 Minimax Probability Machine for binary classification

We will briefly describe the underlying concepts of the MPMC framework as developed by Lanckriet et al. (see [4]). The goal of MPMC is to find a decision boundary $\mathcal{H}(\mathbf{a}, b) = \{\mathbf{z}|\mathbf{a}^T\mathbf{z} = b\}$ such that the minimum probability $\Omega_\mathcal{H}$ of classifying future data correctly is maximized. If we assume that the two classes are generated from random vectors $\mathbf{x}$ and $\mathbf{y}$,

we can express this probability bound just in terms of the means and covariances of these random vectors:

$$\Omega_{\mathcal{H}} = \inf_{\mathbf{x} \sim (\bar{x}, \boldsymbol{\Sigma}_{\mathbf{x}}), \mathbf{y} \sim (\bar{y}, \boldsymbol{\Sigma}_{\mathbf{y}})} Pr\{\mathbf{a}^T \mathbf{x} \geq b \wedge \mathbf{a}^T \mathbf{y} \leq b\} \tag{2}$$

Note that we do not make any distributional assumptions other than that $\bar{\mathbf{x}}, \boldsymbol{\Sigma}_{\mathbf{x}}, \bar{\mathbf{y}}$, and $\boldsymbol{\Sigma}_{\mathbf{x}}$ are bounded. Exploiting a theorem from Marshall and Olkin [1], it is possible to rewrite (2) as a closed form expression:

$$\Omega_{\mathcal{H}} = \frac{1}{1 + m^2} \tag{3}$$

where

$$m = \min_{\mathbf{a}} \sqrt{\mathbf{a}^T \boldsymbol{\Sigma}_{\mathbf{x}} \mathbf{a}} + \sqrt{\mathbf{a}^T \boldsymbol{\Sigma}_{\mathbf{y}} \mathbf{a}} \quad s.t. \quad \mathbf{a}^T (\bar{\mathbf{x}} - \bar{\mathbf{y}}) = 1 \tag{4}$$

The optimal hyperplane parameter $\mathbf{a}_*$ is the vector that minimizes (4). The hyperplane parameter $b_*$ can then be computed as:

$$b_* = \mathbf{a}_*^T \bar{\mathbf{x}} - \frac{\sqrt{\mathbf{a}_*^T \boldsymbol{\Sigma}_{\mathbf{x}} \mathbf{a}_*}}{m} \tag{5}$$

A new data point $z_{new}$ is classified according to $sign(\mathbf{a}_*^T \mathbf{z}_{new} - b_*)$; if this yields $+1$, $z_{new}$ is classified as belonging to class $\mathbf{x}$, otherwise it is classified as belonging to class $\mathbf{y}$.

## 2.2 Sparse MPM classification

One of the appealing properties of Support Vector Machines is that their models typically rely only on a small fraction of the training examples, the so called support vectors. The models obtained from the kernelized MPMC, however, use *all* of the training examples (see [4]), i.e. the decision hyperplane will look like:

$$\sum_{i=1}^{N_x} a_i^{(x)} K(\mathbf{x_i}, \mathbf{z}) + \sum_{i=1}^{N_y} a_i^{(y)} K(\mathbf{y_i}, \mathbf{z}) = b \tag{6}$$

where in general all $a_i^{(x)}, a_i^{(y)} \neq 0$.

This brings up the question whether one can construct sparse models for the MPMC where most of the coefficients $a_i^{(x)}$ or $a_i^{(y)}$ are zero. In this paper we propose to do this by starting with an initially "empty" model and then adding basis functions one by one. As we will see shortly, this approach is speeding up both learning and evaluation time while it is still maintaining the distribution free probability bound of the MPMC.

Before we outline the algorithm we introduce some notation:

$N$ = $N_x + N_y$ the total number of training examples

$\ell$ = $(\ell_1, ..., \ell_N)^T \in \{-1, 1\}^N$ the labels of the training data

$\widehat{\ell}^{(k)}$ = $(\widehat{\ell}_1^{(k)}, ..., \widehat{\ell}_N^{(k)})^T \in \mathbf{R}^N$ output of the model after adding the $k$th basis function

$\mathbf{a}^{(k)}$ = the MPMC hyperplane coefficients when adding the $k$th basis function

$b^{(k)}$ = the MPMC hyperplane offset when adding the $k$th basis function

$\vec{K_b}$ = $(K_{\mathbf{v}}(\mathbf{v}, \mathbf{x_1}), ..., K_{\mathbf{v}}(\mathbf{v}, \mathbf{x_{N_x}}), K_{\mathbf{v}}(\mathbf{v}, \mathbf{y_1}), ..., K_{\mathbf{v}}(\mathbf{v}, \mathbf{y_{N_y}}))^T$ basis function evaluated on all training examples (empirical map)

$\vec{K_{x_v}}$ = $(K_{\mathbf{v}}(\mathbf{v}, \mathbf{x_1}), ..., K_{\mathbf{v}}(\mathbf{v}, \mathbf{x_{N_x}}))^T$ evaluated only on positive examples

$\vec{K_{y_v}}$ = $(K_{\mathbf{v}}(\mathbf{v}, \mathbf{y_1}), ..., K_{\mathbf{v}}(\mathbf{v}, \mathbf{y_{N_y}}))^T$ evaluated only on negative examples

Note that $\widehat{\ell}^{(k)}$ is a vector of real numbers (the distances of the training data to the hyperplane before applying the $sign$ function). $\mathbf{v} \in \mathbf{R}^d$ is the training vector generating the basis function $\vec{K}_{\mathbf{v}}$ [1]. We will simply write $\vec{K}^{(k)}, \vec{K}_{\mathbf{x}}^{(k)}, \vec{K}_{\mathbf{y}}^{(k)}$ for the $k$th basis function.

For the first basis we are solving the one dimensional MPMC:

$$m = \min_{a} \sqrt{a\sigma^2_{\vec{K}^{(1)}_\mathbf{x}}a} + \sqrt{a\sigma^2_{\vec{K}^{(1)}_\mathbf{y}}a} \quad s.t. \quad a(\overline{\vec{K}^{(1)}_\mathbf{x}} - \overline{\vec{K}^{(1)}_\mathbf{y}}) = 1 \tag{7}$$

where $\overline{\vec{K}^{(1)}_\mathbf{x}}$ and $\sigma^2_{\vec{K}^{(1)}_\mathbf{x}}$ are the mean and variance of the vector $\vec{K}^{(1)}_\mathbf{x}$ (which is the first basis function evaluated on all positive training examples).

Because of the constraint the feasible region contains just one value for $a^{(1)}$:

$$a^{(1)} = 1/(\overline{\vec{K}^{(1)}_\mathbf{x}} - \overline{\vec{K}^{(1)}_\mathbf{y}})$$

$$b^{(1)} = a^{(1)}\overline{\vec{K}^{(1)}_\mathbf{x}} - \frac{\sqrt{a\sigma^2_{\vec{K}^{(1)}_\mathbf{x}}a}}{\sqrt{a\sigma^2_{\vec{K}^{(1)}_\mathbf{x}}a} + \sqrt{a\sigma^2_{\vec{K}^{(1)}_\mathbf{y}}a}} = a^{(1)}\overline{\vec{K}^{(1)}_\mathbf{x}} - \frac{\sigma_{\vec{K}^{(1)}_\mathbf{x}}}{\sigma_{\vec{K}^{(1)}_\mathbf{x}} + \sigma_{\vec{K}^{(1)}_\mathbf{y}}} \tag{8}$$

The first model then looks like:

$$\widehat{\ell}^{(1)} = a^{(1)}\vec{K}^{(1)} - b^{(1)} \tag{9}$$

All of the subsequent models use the previous estimation $\widehat{\ell}^{(k)}$ as one input and the next basis $\vec{K}^{(k+1)}$ as the other input. We set up the two dimensional classification problem:

$$\mathbf{x}^{(k+1)} = [\widehat{\ell}^{(k)}_\mathbf{x}, \vec{K}^{(k+1)}_\mathbf{x}] \in \mathbf{R}^{N_x \times 2}$$
$$\mathbf{y}^{(k+1)} = [\widehat{\ell}^{(k)}_\mathbf{y}, \vec{K}^{(k+1)}_\mathbf{y}] \in \mathbf{R}^{N_y \times 2} \tag{10}$$

And solve the following optimization problem:

$$m = \min_{\mathbf{a}} \sqrt{\mathbf{a}^T \Sigma_{\mathbf{x}^{(k+1)}} \mathbf{a}} + \sqrt{\mathbf{a}^T \Sigma_{\mathbf{y}^{(k+1)}} \mathbf{a}} \quad s.t. \quad \mathbf{a}^T(\overline{\mathbf{x}^{(k+1)}} - \overline{\mathbf{y}^{(k+1)}}) = 1 \tag{11}$$

where $\overline{\mathbf{x}^{(k+1)}}$ is the 2-dimensional mean vector $(\overline{\widehat{\ell}^{(k)}_\mathbf{x}}, \overline{\vec{K}^{(k+1)}_\mathbf{x}})^T$ and where $\Sigma_{\mathbf{x}^{(k+1)}}$ is the $2 \times 2$ sample covariance matrix of the vectors $\widehat{\ell}^{(k)}_\mathbf{x}$ and $\vec{K}^{(k+1)}_\mathbf{x}$.

Let $\mathbf{a}^{(k+1)} = (a_1^{(k+1)}, a_2^{(k+1)})^T$ be the optimal solution of (11). We set:

$$b^{(k+1)} = \mathbf{a}^{(k+1)^T}\overline{\mathbf{x}^{(k+1)}} - \frac{\sqrt{\mathbf{a}^{(k+1)^T}\Sigma_{\mathbf{x}^{(k+1)}}\mathbf{a}^{(k+1)}}}{\sqrt{\mathbf{a}^{(k+1)^T}\Sigma_{\mathbf{x}^{(k+1)}}\mathbf{a}^{(k+1)}} + \sqrt{\mathbf{a}^{(k+1)^T}\Sigma_{\mathbf{y}^{(k+1)}}\mathbf{a}^{(k+1)}}} \tag{12}$$

and obtain the next model as:

$$\widehat{\ell}^{(k+1)} = a_1^{(k+1)}\widehat{\ell}^{(k)} + a_2^{(k+1)}\vec{K}^{(k+1)} - b^{(k+1)} \tag{13}$$

As stated above, one computational advantage of SMPMC is that we typically use only a small number of training examples to obtain our final model (i.e. $k << N$). Another benefit is that we have to solve only one and two dimensional MPMC problems. As seen in (8) the one dimensional solution is trivial to compute. An analysis of the two dimensional problem shows that it can be reduced to the problem of finding the roots of a fourth order polynomial. Polynomials of degree 4 still have closed form solutions (see e.g. [6]) which can be computed efficiently. In the standard MPMC algorithm (see [4]), however, the solution $\mathbf{a}$ for equation (4) has $N$ dimensions and can therefore only be found by expensive numerical methods.

It may seem that the values of $\Omega = 1/(1 + m^2)$ which we obtain from (11) are not true for the whole model since we are considering only two dimensional problems and not all of the $k + 1$ dimensions we have added so far through our basis functions. But it turns out that the "local" bound (from the 2D MPMC) is indeed equal to the "global" bound (when considering all $k + 1$ dimensions). We state this fact more formally in the following theorem:

**Theorem 1:** *Let $\widehat{\ell}^{(k)} = c_0 + c_1\vec{K}^{(1)} + ... + c_k\vec{K}^{(k)}$ be the sparse MPMC model at the $k$th iteration ($k \geq 1$) and let $a_1^{(k+1)}, a_2^{(k+1)}, b^{(k+1)}$ be the solution of the two dimensional MPMC: $\widehat{\ell}^{(k+1)} = a_1^{(k+1)}\widehat{\ell}^{(k)} + a_2^{(k+1)}\vec{K}^{(k+1)} - b^{(k+1)}$.*
*Then the values of $\Omega$ for the two dimensional MPMC and for the $k+1$ dimensional MPMC are the same.*

**Proof:** see Appendix

### 2.3 Selection of bases and Gaussian Kernel widths

In our experiments we are using the Gaussian kernel which looks like:

$$K_\sigma(\mathbf{u}, \mathbf{v}) = exp(-\frac{||\mathbf{u} - \mathbf{v}||_2^2}{2\sigma^2}) \qquad (14)$$

where $\sigma$ is the so called kernel width. As mentioned before, one typically has to choose $\sigma$ manually or determine it by cross validation (see [4]). The SMPMC algorithm greedily selects a basis function – out of a randomly chosen *candidate set* – to maximize $\Omega$ which is equivalent to minimizing the value of $m$ in (7) and (11). Before we state the optimization problem for the one and two dimensional MPMC we rewrite (14) so that we can get rid of the denominator:

$$K_\gamma(\mathbf{u}, \mathbf{v}) = exp(-\gamma||\mathbf{u} - \mathbf{v}||_2^2) \quad \gamma \geq 0 \qquad (15)$$

The optimization problem we solve for the first iteration is then:

$$\min_\gamma m(\gamma) = \min_a \sqrt{a\sigma^2_{\vec{K}_{\mathbf{x}}^{(1)}}a} + \sqrt{a\sigma^2_{\vec{K}_{\mathbf{y}}^{(1)}}a} \quad s.t. \quad a(\overrightarrow{K_{\mathbf{x}}^{(1)}} - \overrightarrow{K_{\mathbf{y}}^{(1)}}) = 1 \qquad (16)$$

note that – even though we did not state it explicitly – the statistics $\sigma^2_{\vec{K}_{\mathbf{x}}^{(1)}}, \sigma^2_{\vec{K}_{\mathbf{y}}^{(1)}}, \overrightarrow{K_{\mathbf{x}}^{(1)}}$, and $\overrightarrow{K_{\mathbf{y}}^{(1)}}$ (and consequently the coefficient $a$) all depend on the kernel parameter $\gamma$.

The two dimensional problem that has to be solved for all subsequent iterations $k \geq 2$ turns into the following optimization problem for $\gamma$:

$$\min_\gamma m(\gamma) = \min_\mathbf{a} \sqrt{\mathbf{a}^T\Sigma_{\mathbf{x}^{(k+1)}}\mathbf{a}} + \sqrt{\mathbf{a}^T\Sigma_{\mathbf{y}^{(k+1)}}\mathbf{a}} \quad s.t. \quad \mathbf{a}^T(\overline{\mathbf{x}^{(k+1)}} - \overline{\mathbf{y}^{(k+1)}}) = 1 \quad (17)$$

Again, $\overline{\mathbf{x}^{(k+1)}}, \overline{\mathbf{y}^{(k+1)}}, \Sigma_{\mathbf{x}^{(k+1)}}$, and $\Sigma_{\mathbf{y}^{(k+1)}}$ all depend on the kernel parameter $\gamma$ and from these four statistics we can compute the minimizer $\mathbf{a} \in \mathbf{R}^2$ analytically.

### 2.4 Feature selection

For doing feature selection with Gaussian kernels one has to replace the uniform kernel width $\gamma$ with a $d$ dimensional vector $\vec{\gamma}$ of kernel weightings:

$$K_{\vec{\gamma}}(\mathbf{u}, \mathbf{v}) = exp(-\sum_{l=1}^d \gamma_l(u_l - v_l)^2) \quad (\gamma_l \geq 0 \quad l = 1, ..., d) \qquad (18)$$

Note that the optimization problems (16) and (17) for the one respectively two dimensional MPMC are now $d$ dimensional instead of just one dimensional.

## 3 Experiments

In this section we describe the results we obtained for SMPMC on various classification benchmarks. We used the same data sets as Lanckriet et al. in [4] for the standard MPMC. The data sets were randomly divided into 90% training data and 10% test data and the results were averaged over 50 runs for each of the five problems (see table 1). In all the experiments listed in table 1 we used the feature selection algorithm (with the exception of Sonar where width selection was used) and had a candidate set of size 5, i.e. at each iteration the best basis out of 5 randomly chosen candidates was selected. The results we obtained are comparable to the ones reported by Lanckriet et al [4]. Note that for all of the data sets SMPMC uses significantly less basis functions than MPMC does which directly translates into an accordingly smaller evaluation cost. The differences in training cost are shown in table 2. The total training time for standard MPMC takes into account the 50-fold cross validation and 10 candidates for the kernel parameter. We observe that for all of the five data sets the training cost of sparse MPMC is only a fraction of the one for standard MPMC.

The two plots in figure 1 show what typical learning curves for sparse MPMC look like. As the number of basis function increases, both the bound $\Omega$ and the test set accuracy start

Table 1: Bound $\Omega$, Test set accuracy (TSA), number of bases (B) for sparse and standard MPMC

| Dataset | SMPMC | | | Standard MPMC (Lanckriet et al.) | | |
|---|---|---|---|---|---|---|
| | $\Omega$ | TSA | B | $\Omega$ | TSA | B |
| Twonorm | $86.4 \pm 0.1\%$ | $98.3 \pm 0.4\%$ | 25 | $91.3 \pm 0.1\%$ | $95.7 \pm 0.5\%$ | 270 |
| Breast Cancer | $90.9 \pm 0.1\%$ | $96.8 \pm 0.3\%$ | 50 | $89.1 \pm 0.1\%$ | $96.9 \pm 0.3\%$ | 614 |
| Ionosphere | $77.7 \pm 0.2\%$ | $91.6 \pm 0.5\%$ | 25 | $89.3 \pm 0.2\%$ | $91.5 \pm 0.7\%$ | 315 |
| Pima Diabetes | $38.2 \pm 0.1\%$ | $75.4 \pm 0.7\%$ | 50 | $32.5 \pm 0.2\%$ | $76.2 \pm 0.6\%$ | 691 |
| Sonar | $78.5 \pm 0.2\%$ | $86.4 \pm 1.0\%$ | 80 | $99.9 \pm 0.1\%$ | $87.5 \pm 0.9\%$ | 187 |

Table 2: training time (in seconds) for Matlab implementations of SMPMC and MPMC

| Dataset | # training examples | SMPMC training time | Standard MPMC (Lanckriet et al.) | |
|---|---|---|---|---|
| | | | one optimization | total training time |
| Twonorm | 270 | 125.0 | 23.9 | 1199.2 |
| Breast Cancer | 614 | 188.5 | 122.4 | 6123.2 |
| Ionosphere | 315 | 416.3 | 28.1 | 1404.3 |
| Pima Diabetes | 691 | 165.6 | 186.5 | 9324.2 |
| Sonar | 187 | 35.3 | 8.7 | 435.1 |

to go up and after a while stabilize. The stabilization point usually occurs earlier when one does full feature selection (a $\gamma$ weight for each input dimension) instead of kernel width selection (one uniform $\gamma$ for all dimensions). We also experimented with different sizes for the candidate set. The plots in figure 2 show what happens for 1, 5, and 10 candidates. The overall behavior is that the test set accuracy as well as the $\Omega$ value converge earlier for larger candidate sets (but note that a larger candidate set also increases the computational cost per iteration).

As seen in figure 1, feature selection gives usually better results in terms of the bound $\Omega$ and the test set accuracy. Furthermore, a feature selection algorithm should indicate which features are relevant and which are not. We set up an experiment for the Twonorm data (which has 20 input features) where we added 20 additional noisy features that were not related to the output. The results are shown in figure 3 and demonstrate that the feature selection algorithm obtained from SMPMC is able to distinguish between relevant and irrelevant features.

## 4 Conclusion & future work

This paper introduces a new algorithm (Sparse Minimax Probability Machine Classification - SMPMC) for building sparse classification models that provide a lower bound on the probability of classifying future data correctly. We have shown that the method of iteratively adding basis functions has significant computational advantages over the standard MPMC, while it still maintains the distribution free probability bound $\Omega$. Experimental results indicate that automated selection of kernel parameters, as well as automated feature selection (weighting), both key characteristics of the SMPMC algorithm, result in error rates that are competitive with those obtained by models where these parameters must be tuned by computationally expensive cross validation.

Future research on sparse greedy MPMC will focus on establishing a theoretical framework for a stopping criterion, when adding more basis functions (kernels) will not significantly reduce error rates, and may lead to overfitting. Also, experiments have so far focused on using Gaussian kernels as basis functions. From the experience with other kernel algorithms, it is known that other type of kernels (polynomial, tanh) can yield better results for certain applications. Furthermore, our framework is not limited to Mercer kernels, and other types

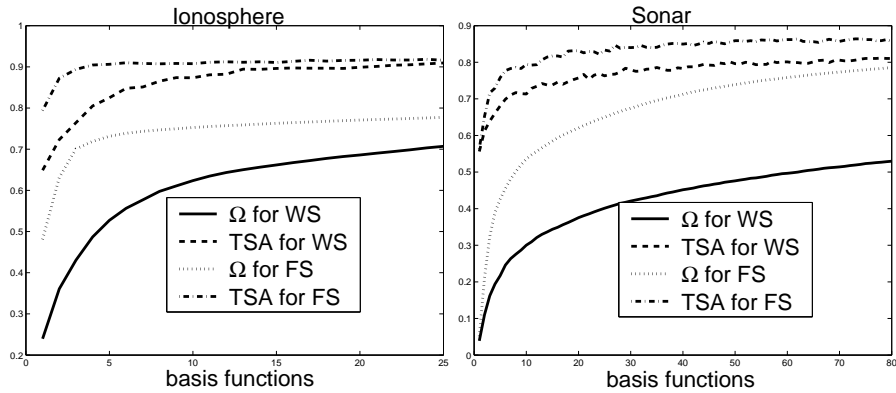

Figure 1: Bound $\Omega$ and Test Set accuracy (TSA) for width selection (WS) and feature selection (FS). Note that the accuracies are all higher than the corresponding bounds.

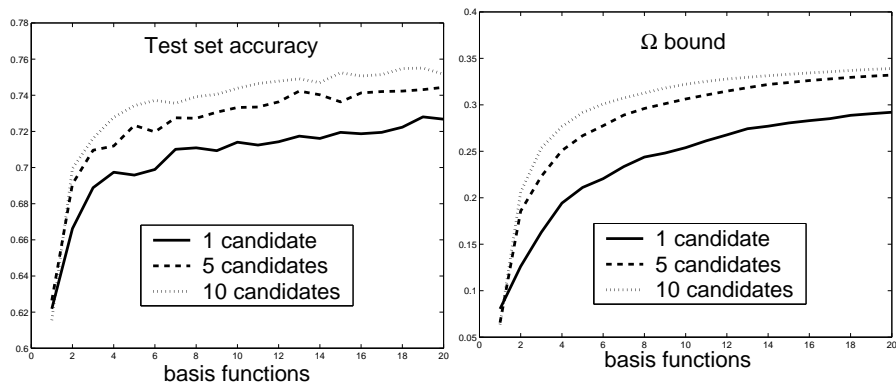

Figure 2: Accuracy and bound for the Diabetes data set using 1,5 or 10 basis candidates per iteration. Again, the $\Omega$ bound is a true lower bound on the test set accuracy.

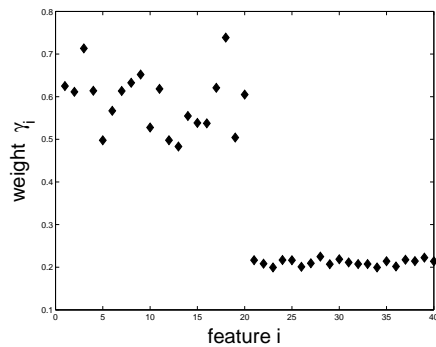

Figure 3: Average feature weighting for the Twonorm data set over 50 test runs. The first 20 features are the original inputs, the last 20 features are additional noisy inputs

of basis functions are also worth investigating. Recent work by Crammer et al. [7] uses boosting to construct a suitable kernel matrix iteratively. An interesting open question is how this approach relates to sparse greedy MPMC.

## Footnotes

[1]Note that we use the same symbol $\vec{K}$ for both the empirical map and the induced function. It will always be clear from the context what $\vec{K}$ refers to.

# References

[1] A. W. Marshall and I. Olkin. Multivariate chebyshev inequalities. *Annals of Mathematical Statistics*, 31(4):1001–1014, 1960.

[2] I. Popescu and D. Bertsimas. Optimal inequalities in probability theory: A convex optimization approach. Technical Report TM62, INSEAD, Dept. Math. O.R., Cambridge, Mass, 2001.

[3] G. R. G. Lanckriet, L. E. Ghaoui, C. Bhattacharyya, and M. I. Jordan. Minimax probability machine. In T. G. Dietterich, S. Becker, and Z. Ghahramani, editors, *Advances in Neural Information Processing Systems 14*, Cambridge, MA, 2002. MIT Press.

[4] G. R. G. Lanckriet, L. E. Ghaoui, C. Bhattacharyya, and M. I. Jordan. A robust minimax approach to classification. *Journal of Machine Learning Research*, 3:555–582, 2002.

[5] B. Schölkopf and A. Smola. *Learning with Kernels*. MIT Press, Cambridge, MA, 2002.

[6] William H. Beyer. *CRC Standard Mathemathical Tables*, page 12. CRC Press Inc., Boca Raton, FL, 1987.

[7] K. Crammer, J. Keshet, and Y. Singer. Kernel design using boosting. In T. G. Dietterich, S. Becker, and Z. Ghahramani, editors, *Advances in Neural Information Processing Systems 15*, Cambridge, MA, 2003. MIT Press.

## Appendix: Proof of Theorem 1

We have to show that the values of $m$ are equal for the two dimensional MPMC and the $k + 1$ dimensional MPMC. We will just show the equivalence for the first term $\sqrt{\mathbf{a}^T \Sigma_{\mathbf{x}} \mathbf{a}}$, an analogue argumentation will hold for the second term.

For the two dimensional MPMC we have the following for the term under the square root:

$$
\begin{pmatrix} a_1^{(k+1)} & a_2^{(k+1)} \end{pmatrix} \begin{pmatrix} \sigma_{\widehat{\ell}_{\mathbf{x}}^{(k)}}^2 & \sigma_{\widehat{\ell}_{\mathbf{x}}^{(k)} \vec{K}_{\mathbf{x}}^{(k+1)}} \\ \sigma_{\vec{K}_{\mathbf{x}}^{(k+1)} \widehat{\ell}_{\mathbf{x}}^{(k)}} & \sigma_{\vec{K}_{\mathbf{x}}^{(k+1)}}^2 \end{pmatrix} \begin{pmatrix} a_1^{(k+1)} \\ a_2^{(k+1)} \end{pmatrix}
\tag{19}
$$
$$
= [a_1^{(k+1)}]^2 \sigma_{\widehat{\ell}_{\mathbf{x}}^{(k)}}^2 + 2 a_1^{(k+1)} a_2^{(k+1)} \sigma_{\widehat{\ell}_{\mathbf{x}}^{(k)} \vec{K}_{\mathbf{x}}^{(k+1)}} + [a_2^{(k+1)}]^2 \sigma_{\vec{K}_{\mathbf{x}}^{(k+1)}}^2
$$

Note that we can rewrite

$$
\begin{aligned}
\sigma_{\widehat{\ell}_{\mathbf{x}}^{(k)}}^2 &= Cov(c_0 + c_1 \vec{K}_{\mathbf{x}}^{(1)} + ... + c_k \vec{K}_{\mathbf{x}}^{(k)}, c_0 + c_1 \vec{K}_{\mathbf{x}}^{(1)} + ... + c_k \vec{K}_{\mathbf{x}}^{(k)}) \\
&= \sum_{i=1}^{k} \sum_{j=1}^{k} c_i c_j Cov(\vec{K}_{\mathbf{x}}^{(i)}, \vec{K}_{\mathbf{x}}^{(j)}) \\
\sigma_{\widehat{\ell}_{\mathbf{x}}^{(k)} \vec{K}_{\mathbf{x}}^{(k+1)}} &= Cov(c_0 + c_1 \vec{K}_{\mathbf{x}}^{(1)} + ... + c_k \vec{K}_{\mathbf{x}}^{(k)}, \vec{K}_{\mathbf{x}}^{(k+1)}) \\
&= \sum_{i=1}^{k} c_i Cov(\vec{K}_{\mathbf{x}}^{(i)}, \vec{K}_{\mathbf{x}}^{(k+1)})
\end{aligned}
\tag{20}
$$

by using properties of the sample covariance (linearity, $Cov(const, X) = 0$).

For the $k + 1$ dimensional MPMC let us first determine the $k + 1$ coefficients:

$$
\begin{aligned}
\widehat{\ell}^{(k+1)} &= a_1^{(k+1)}(c_0 + c_1 \vec{K}_{\mathbf{x}}^{(1)} + ... + c_k \vec{K}_{\mathbf{x}}^{(k)}) + a_2^{(k+1)} \vec{K}_{\mathbf{x}}^{(k+1)} - b^{(k+1)} \\
&= a_1^{(k+1)} c_1 \vec{K}_{\mathbf{x}}^{(1)} + ... + a_1^{(k+1)} c_k \vec{K}_{\mathbf{x}}^{(k)} + a_2^{(k+1)} \vec{K}_{\mathbf{x}}^{(k+1)} + a_1^{(k+1)} c_0 - b^{(k+1)}
\end{aligned}
$$

The term under the square root then looks like:

$$
\begin{pmatrix} a_1^{(k+1)} c_1 \\ ... \\ a_1^{(k+1)} c_k \\ a_2^{(k+1)} \end{pmatrix}^T \begin{pmatrix} \sigma_{\vec{K}_{\mathbf{x}}^{(1)}}^2 & ... & \sigma_{\vec{K}_{\mathbf{x}}^{(1)} \vec{K}_{\mathbf{x}}^{(k)}} & \sigma_{\vec{K}_{\mathbf{x}}^{(1)} \vec{K}_{\mathbf{x}}^{(k+1)}} \\ ... & ... & ... & ... \\ \sigma_{\vec{K}_{\mathbf{x}}^{(k)} \vec{K}_{\mathbf{x}}^{(1)}} & ... & \sigma_{\vec{K}_{\mathbf{x}}^{(k)}}^2 & \sigma_{\vec{K}_{\mathbf{x}}^{(k)} \vec{K}_{\mathbf{x}}^{(k+1)}} \\ \sigma_{\vec{K}_{\mathbf{x}}^{(k+1)} \vec{K}_{\mathbf{x}}^{(1)}} & ... & \sigma_{\vec{K}_{\mathbf{x}}^{(k+1)} \vec{K}_{\mathbf{x}}^{(k)}} & \sigma_{\vec{K}_{\mathbf{x}}^{(k+1)}}^2 \end{pmatrix} \begin{pmatrix} a_1^{(k+1)} c_1 \\ ... \\ a_1^{(k+1)} c_k \\ a_2^{(k+1)} \end{pmatrix}
\tag{21}
$$

Multiplying out (21) and substituting according to the equations in (20) yields exactly expression (19) (which is the $\mathbf{a}^T \Sigma_{\mathbf{x}} \mathbf{a}$ term of the two dimensional MPM). Since this equivalence will hold likewise for the $\sqrt{\mathbf{a}^T \Sigma_{\mathbf{y}} \mathbf{a}}$ term in $m$, we have shown that $m$ (and therefore $\Omega$) is equal for the two dimensional and the $k + 1$ dimensional MPMC. $\qquad \square$